# Learning Sparse Image Codes using a Wavelet Pyramid Architecture

**Bruno A. Olshausen**
Department of Psychology and
Center for Neuroscience, UC Davis
1544 Newton Ct.
Davis, CA 95616
*baolshausen@ucdavis.edu*

**Phil Sallee**
Department of Computer Science
UC Davis
Davis, CA 95616
*sallee@cs.ucdavis.edu*

**Michael S. Lewicki**
Department of Computer Science and
Center for the Neural Basis of Cognition
Carnegie Mellon University
Pittsburgh, PA 15213
*lewicki@cnbc.cmu.edu*

## Abstract

We show how a wavelet basis may be adapted to best represent natural images in terms of sparse coefficients. The wavelet basis, which may be either complete or overcomplete, is specified by a small number of spatial functions which are repeated across space and combined in a recursive fashion so as to be self-similar across scale. These functions are adapted to minimize the estimated code length under a model that assumes images are composed of a linear superposition of sparse, independent components. When adapted to natural images, the wavelet bases take on different orientations and they evenly tile the orientation domain, in stark contrast to the standard, non-oriented wavelet bases used in image compression. When the basis set is allowed to be overcomplete, it also yields higher coding efficiency than standard wavelet bases.

## 1 Introduction

The general problem we address here is that of learning efficient codes for representing natural images. Our previous work in this area has focussed on learning basis functions that represent images in terms of sparse, independent components [1, 2]. This is done within the context of a linear generative model for images, in which an image $I(x,y)$ is described in terms of a linear superposition of basis functions $b_i(x,y)$ with amplitudes $a_i$, plus noise $\nu(x,y)$:

$$I(x,y) = \sum_i a_i \, b_i(x,y) + \nu(x,y) \qquad (1)$$

A sparse, factorial prior is imposed upon the coefficients $a_i$, and the basis functions are adapted so as to maximize the average log-probability of images under the model (which is equivalent to minimizing the model's estimate of the code length of the images). When the model is trained on an ensemble of whitened natural images, the basis functions converge to a set of spatially localized, oriented, and bandpass functions that tile the joint space of position and spatial-frequency in a manner similar to a wavelet basis. Similar results have been achieved using other forms of independent components analysis [3, 4].

One of the disadvantages of this approach, from an image coding perspective, is that it may only be applied to small sub-images (e.g., 12 × 12 pixels) extracted from a larger image. Thus, if an image were to be coded using this method, it would need to be blocked and would thus likely introduce blocking artifacts as the result of quantization or sparsification of the coefficients. In addition, the model is unable to capture spatial structure in the images that is larger than the image block, and scaling up the algorithm to significantly larger blocks is computationally intractable.

The solution to these problems that we propose here is to assume translation- and scale-invariance among the basis functions, as in a wavelet pyramid architecture. That is, if a basis function is learned at one position and scale, then it is assumed to be repeated at all positions (spaced apart by two positions horizontally and vertically) and scales (in octave increments). Thus, the entire set of basis functions for tiling a large image may be learned by adapting only a handful of parameters— i.e., the wavelet filters and the scaling function that is used to expand them across scale.

We show here that when a wavelet image model is adapted to natural images to yield coefficients that are sparse and as statistically independent as possible, the wavelet functions converge to a set of oriented functions, and the scaling function converges to a circularly symmetric lowpass filter appropriate for generating self-similarity across scale. Moreover, the resulting coefficients achieve higher coding efficiency (higher SNR for a fixed bit-rate) than traditional wavelet bases which are typically designed "by hand" according to certain mathematical desiderata [5].

## 2 Wavelet image model

The wavelet image model is specified by a relatively small number of parameters, consisting of a set of wavelet functions $\psi_i(x,y)$, $i = 1..M$, and scaling function $\phi(x,y)$. An image is generated by upsampling and convolving the coefficients at a given band $i$ with $\psi_i$ (or with $\phi$ at the lowest-resolution level of the pyramid), followed by successive upsampling and convolution with $\phi$, depending on their level within the pyramid. The wavelet image model for an $L$ level pyramid is specified mathematically as

$$I(x,y) = g(x,y,0) + \nu(x,y) \tag{2}$$

$$g(x,y,l) = \begin{cases} a^{L-1}(x,y) & l = L-1 \\ I^l(x,y) & l < L-1 \end{cases} \tag{3}$$

$$I^l(x,y) = [g(x,y,l+1)\uparrow 2] * \phi(x,y) + \sum_{i=1}^{M} [a_i^l(x,y)\uparrow 2] * \psi_i(x,y) \tag{4}$$

where the coefficients $a$ are indexed by their position, $x,y$, band, $i$, and level of resolution $l$ within the pyramid ($l = 0$ is the highest resolution level). The symbol

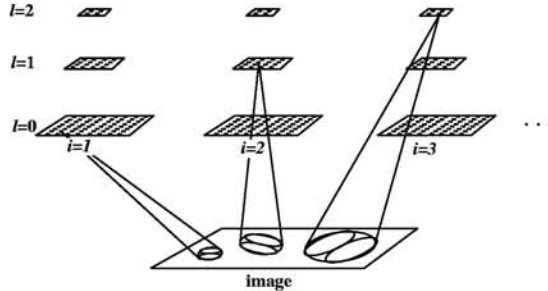

Figure 1: Wavelet image model. Shown are the coefficients of the first three levels of a pyramid ($l = 0, 1, 2$), with each level split into a number of different bands ($i = 1...M$). The highest level ($l = 3$) is not shown and contains only one lowpass band.

$\uparrow 2$ denotes upsampling by two and is defined as

$$f(x,y)\uparrow 2 \; \equiv \; \begin{cases} f(\frac{x}{2},\frac{y}{2}) & x\text{ even } \& \; y\text{ even} \\ 0 & \text{otherwise} \end{cases} \qquad (5)$$

The wavelet pyramid model is schematically illustrated in figure 1. Traditional wavelet bases typically utilize three bands ($M = 3$), in which case the representation is *critically sampled* (same number of coefficients as image pixels). Here, we shall also examine the cases of $M = 4$ and 6, in which the representation is *overcomplete* (more coefficients than image pixels).

Because the image model is linear, it may be expressed compactly in vector/matrix notation as

$$\mathbf{I} = \mathbf{G}\,\mathbf{a} + \boldsymbol{\nu} \qquad (6)$$

where the vector $\mathbf{a}$ is the entire list of coefficient values at all positions, bands, and levels of the pyramid, and the columns of $\mathbf{G}$ are the basis functions corresponding to each coefficient, which are parameterized by $\psi$ and $\phi$. The probability of generating an image $\mathbf{I}$ given a specific state of the coefficients $\mathbf{a}$ and assuming Gaussian i.i.d. noise $\boldsymbol{\nu}$ is then

$$P(\mathbf{I}|\mathbf{a},\theta) = \frac{1}{Z_{\lambda_N}}e^{-\frac{\lambda_N}{2}|\mathbf{I}-\mathbf{G}\,\mathbf{a}|^2} \qquad (7)$$

where $\theta$ denotes the parameters of the model and includes the wavelet pyramid functions $\psi_i$ and $\phi$, as well as the noise variance, $\sigma_\nu^2 = 1/\lambda_N$.

The prior probability distribution over the coefficients is assumed to be factorial and sparse:

$$P(\mathbf{a}) \;=\; \prod_i P(a_i) \qquad (8)$$

$$P(a_i) \;=\; \frac{1}{Z_S}e^{-S(a_i)} \qquad (9)$$

where $S$ is a non-convex function that shapes $P(a_i)$ to have the requisite "sparse" form—i.e., peaked at zero with heavy tails, or positive kurtosis. We choose here $S(x) = \beta\log(1 + (x/\sigma)^2)$, which corresponds to a Cauchy-like prior over the coefficients (an exact Cauchy distribution would be obtained for $\beta = 1$).[1]

# 3 Inferring the coefficients

The coefficients for a particular image are determined by finding the maximum of the posterior distribution (MAP estimate)

$$
\begin{aligned}
\hat{\mathbf{a}} &= \arg\max_{\mathbf{a}} P(\mathbf{a}|\mathbf{I}, \theta) \\
&= \arg\max_{\mathbf{a}} P(\mathbf{I}|\mathbf{a}, \theta) P(\mathbf{a}|\theta) \tag{10} \\
&= \arg\min_{\mathbf{a}} \left[ \frac{\lambda_N}{2} |\mathbf{I} - \mathbf{G}\mathbf{a}|^2 + \sum_i S(a_i) \right] \tag{11}
\end{aligned}
$$

A local minimum may be found via gradient descent, yielding the differential equation

$$
\begin{aligned}
\dot{\mathbf{a}} &\propto \lambda_N \mathbf{G}^T \mathbf{e} - S(\mathbf{a}) \tag{12} \\
\mathbf{e} &= \mathbf{I} - \mathbf{G}\mathbf{a} . \tag{13}
\end{aligned}
$$

The computations involving $\mathbf{G}^T \mathbf{e}$ and $\mathbf{G}\mathbf{a}$ in equations 12 and 13 may be performed quickly and efficiently using fast algorithms for building pyramids and reconstructing from pyramids [7].

# 4 Learning

Our goal in adapting the wavelet model to natural images is to find the functions $\psi_i$ and $\phi$ that minimize the description length $\mathcal{L}$ of images under the model

$$
\begin{aligned}
\mathcal{L} &= -\langle \log P(\mathbf{I}|\theta) \rangle \tag{14} \\
P(\mathbf{I}|\theta) &= \int P(\mathbf{I}|\mathbf{a}, \theta) P(\mathbf{a}|\theta) \, d\mathbf{a} \tag{15}
\end{aligned}
$$

A learning rule for the basis functions may be derived by gradient descent on $\mathcal{L}$:

$$
\begin{aligned}
\Delta\theta_i &\propto -\frac{\partial \mathcal{L}}{\partial \theta_i} \\
&= \lambda_N \left\langle \langle \mathbf{e}^T \frac{\partial \mathbf{G}}{\partial \theta_i} \mathbf{a} \rangle_{P(\mathbf{a}|\mathbf{I}, \theta)} \right\rangle . \tag{16}
\end{aligned}
$$

Instead of sampling from the full posterior distribution, however, we utilize a simpler approximation in which a single sample is taken at the posterior maximum, and so we have

$$
\Delta\theta_i \propto \langle \hat{\mathbf{e}}^T \frac{\partial \mathbf{G}}{\partial \theta_i} \hat{\mathbf{a}} \rangle \tag{17}
$$

where $\hat{\mathbf{e}} = \mathbf{I} - \mathbf{G}\hat{\mathbf{a}}$. The price we pay for this approximation, though, is that the basis functions will grow without bound, since the greater their norm, $|\mathbf{G}_k|$, the smaller each $a_k$ will become, thus decreasing the sparseness penalty in (11). This trivial solution is avoided by adaptively rescaling the basis functions after each learning step so that a target variance on the coefficients is met, as described in an earlier paper [1].

The update rules for $\psi_i$ and $\phi$ are then derived from (17), and may be expressed in terms of the following recursive formulas:

$$
\begin{aligned}
\Delta\psi_i(m, n) &= F_\psi(e(x, y), m, n, 0) \tag{18} \\
F_\psi(f, m, n, l) &\equiv \sum_{x, y} f(2x + m, 2y + n) a_i^l(x, y) + F_\psi([f \star \phi]{\downarrow} 2, m, n, l + 1)
\end{aligned}
$$

$$\Delta\phi(m,n) = F_\phi(e(x,y),m,n,0) \tag{19}$$

$$F_\phi(f,m,n,l) \equiv \sum_{x,y} f(2x+m,2y+n)\,g(x,y,l+1) + F_\phi([f\star\phi]\!\downarrow 2,m,n,l+1)$$

where $\star$ denotes cross-correlation and $\downarrow 2$ denotes downsampling by two. These computations may also be performed efficiently using fast algorithms for building and reconstructing from pyramids [7].

## 5 Results

The image model was trained on a set of 10, pre-whitened $512 \times 512$ natural images that were used in previous studies [1]. The basis function parameters $\psi_i$ and $\phi$ were represented as $5 \times 5$ pixel masks, and were initialized to random numbers. For each update, an $80 \times 80$ subimage was randomly extracted from one of the images, and the coefficients were computed iteratively via (12,13) until the decrease in the energy function was less than 0.1%. The resulting residual, $\mathbf{e}$, was then used for updating the functions $\psi_i$ and $\phi$ according to (18) and (19). The noise parameter $\lambda_N$ was set to 400, corresponding to a noise variance that is 2.5% of the image variance ($\sigma_I^2 = 0.1$). At this level of noise, the image reconstructions are visually indistinguishable from the original. The parameters of the prior used were $\beta = 2.5$, $\sigma = 0.3$. A stable solution began to emerge after about one hour of training for M=3, and after several hours for $M = 6$ (Pentium II, 450 MHz).

Shown in figure 2 are the basis functions learned for the cases $M = 3$, 4 and 6, along with a standard bi-orthogonal 9/7 wavelet (FBI fingerprint standard [8]) for comparison. The difference between the learned wavelets and the standard wavelet is striking, in that the learned wavelets tile the orientation domain more evenly. They also exhibit self-similarity in orientation—i.e., they appear to be rotated versions of one another. Increasing the number of bands $M$ from three to four produces narrower orientation tuning, but increasing overcompleteness beyond that point does not, as shown in the tiling diagram of figure 3. All the learned basis function spectra lie well within the Nyquist bounding box in the 2D Fourier plane, matching the power spectrum of the images in the training set.

Coding efficiency was evaluated by compressing the sparsified coefficients $\hat{\mathbf{a}}$ using the embedded wavelet zerotree encoder [9] and measuring the signal-to-noise ratio for a fixed bit rate (SNR $= 10\log_{10}\sigma_I^2/$mse). The results, shown in table 1, demonstrate that the overcomplete bases ($M = 4$) achieve higher SNR than either of two standard wavelet bases for the same bit rate. Note however that at these levels of SNR the reconstructions are visually identical to the original. At higher compression ratios the learned bases loose their advantage, most likely due to the fact that they are non-orthogonal and hence produce more errors in the reconstruction when the coefficients are quantized.

Table 1: Coding efficiency.

| basis set | SNR |
|---|---|
| $M = 3$ (learned) | 11.2 |
| $M = 4$ (learned) | 11.9 |
| Daubechies 6 | 11.2 |
| FBI 9/7 | 11.4 |

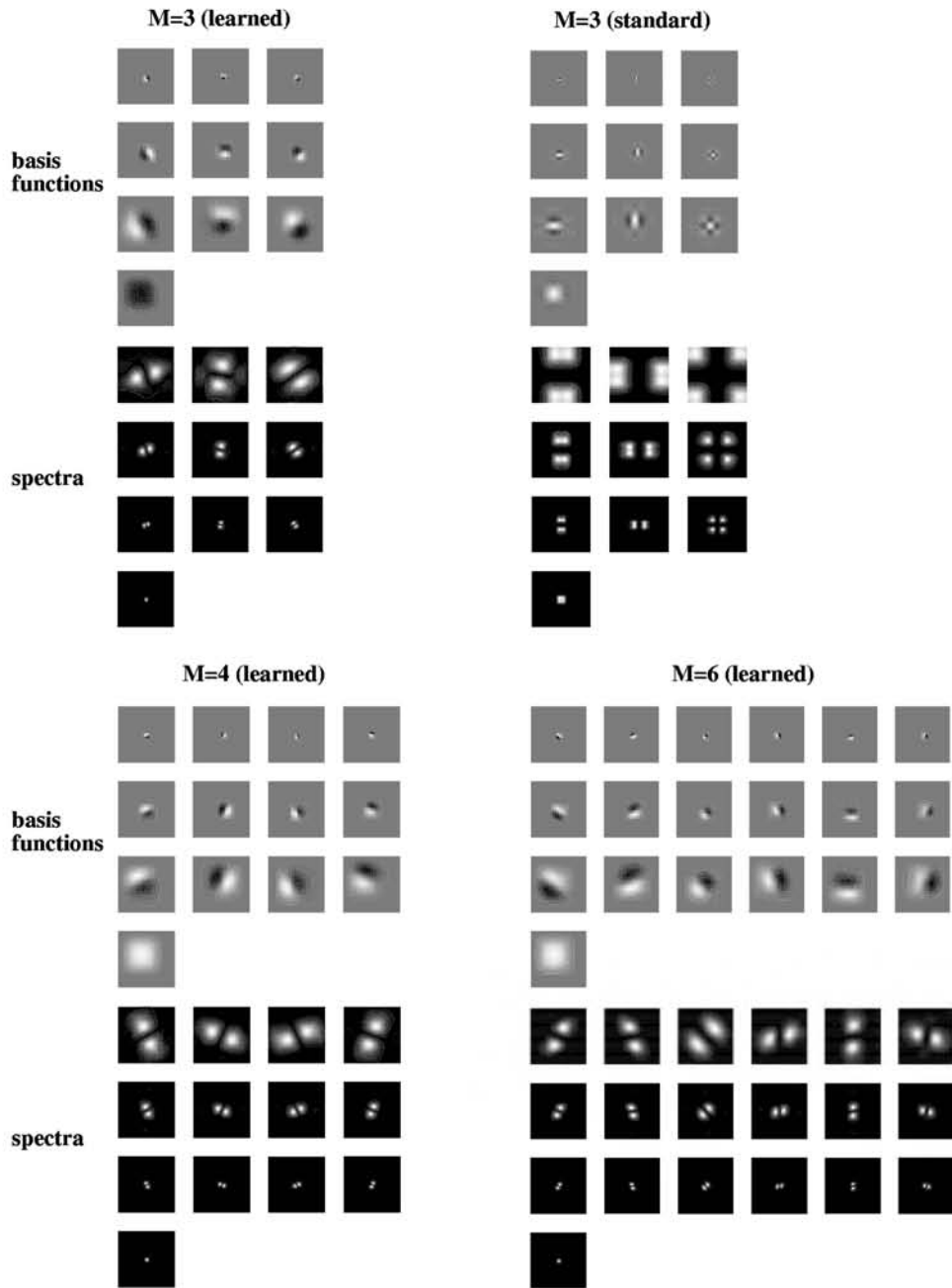

Figure 2: Basis functions and corresponding power spectra for $M = 3$, 4 and 6, along with a standard 9/7 biorthogonal wavelet. Each column shows a different band, while each row shows a different level. The lone basis function in the last row is the scaling function (twice convolved with itself). The power spectra are plotted in the 2D-Fourier plane (vertical vs. horizontal spatial-frequency) with the maximum spatial-frequency at the Nyquist rate.

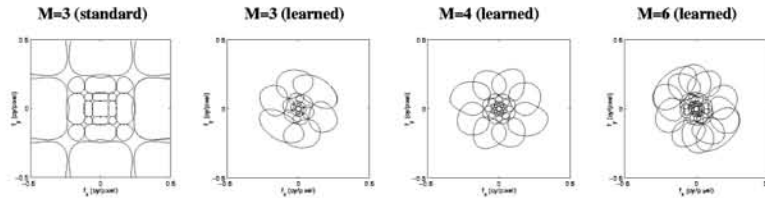

Figure 3: Frequency domain tiling properties. Shown are iso-power contours at 50% of the maximum for each band and level.

# 6   Conclusion

We have shown in this work how a wavelet basis may be adapted so as to represent the structures in natural images in terms of sparse, independent components. Importantly, the algorithm has the capacity to learn overcomplete basis sets, which are capable of tiling the joint space of position, orientation, and spatial-frequency in a more continuous fashion than traditional, critically sampled basis sets [10]. The overcomplete bases exhibit superior coding efficiency, in the sense of achieving higher SNR for a fixed bit rate. Although the improvements in coding efficiency are modest, we believe the method described here has the potential to yield even greater improvements when adapted to more specific image ensembles such as textures.

### Acknowledgments

This work benefited from extensive use of Eero Simoncelli's Matlab pyramid toolbox. Supported by NIMH R29-MH057921.

## Footnotes

[1]A more optimal choice for the prior would be to use a mixture-of-Gaussians distribution, which better captures the sharp peak at zero characteristic of a sparse representation. But properly maximizing the posterior with such a prior presents formidable challenges [6].

# References

[1] Olshausen BA, Field DJ (1997) Sparse coding with an overcomplete basis set: A strategy employed by V1? *Vision Research, 37*: 3311-3325.

[2] Lewicki MS, Olshausen BA (1999) A probabilistic framework for the adaptation and comparison of image codes, *J. Opt. Soc. of Am., A, 16(7)*: 1587-1601.

[3] Bell AJ, Sejnowski TJ (1997) The independent components of natural images are edge filters, *Vision Research, 37*: 3327-3338.

[4] van Hateren JH, van der Schaaff A (1997) Independent component filters of natural images compared with simple cells in primary visual cortex, *Proc. Royal Soc. Lond. B, 265*: 359-366.

[5] Mallat S (1999) *A wavelet tour of signal processing.* Academic Press.

[6] Olshausen BA, Millman KJ (2000). Learning sparse codes with a mixture-of-Gaussians prior. In: *Advances in Neural Information Processing Systems, 12*, S.A. Solla, T.K. Leen, K.R. Muller, eds. MIT Press, pp. 841-847.

[7] Simoncelli EP, Matlab pyramid toolbox.
ftp://ftp.cis.upenn.edu/pub/eero/matlabPyrTools.tar.gz

[8] The Bath Wavelet Warehouse
http://dmsun4.bath.ac.uk/wavelets/warehouse.html

[9] Shapiro JM (1993). Embedded image coding using zerotrees of wavelet coefficients. *IEEE Transactions on Signal Processing, 41(12)*: 3445-3462.

[10] Simoncelli EP, Freeman WT, Adelson EH, Heeger DJ (1992) Shiftable multiscale transforms, *IEEE Transactions on Information Theory, 38(2)*: 587-607.
